# Adaptive Development of Connectionist Decoders for Complex Error-Correcting Codes

**Sheri L. Gish**     **Mario Blaum**
IBM Research Division
Almaden Research Center
650 Harry Road
San Jose, CA 95120

## Abstract

We present an approach for development of a decoder for any complex binary error-correcting code (ECC) via training from examples of decoded received words. Our decoder is a connectionist architecture. We describe two separate solutions: A system-level solution (the Cascaded Networks Decoder); and the ECC-Enhanced Decoder, a solution which simplifies the mapping problem which must be solved for decoding. Although both solutions meet our basic approach constraint for simplicity and compactness, only the ECC-Enhanced Decoder meets our second basic constraint of being a generic solution.

## 1 INTRODUCTION

### 1.1 THE DECODING PROBLEM

An error-correcting code (ECC) is used to identify and correct errors in a received binary vector which is possibly corrupted due to transmission across a noisy channel. In order to use a selected error-correcting code, the *information bits*, or the bits containing the message, are *encoded* into a valid ECC codeword by the addition of a set of extra bits, the *redundancy*, determined by the properties of the selected ECC. To decode a received word, there is a pre-processing step first in which a *syndrome* is calculated from the word. The syndrome is a vector whose length is equal to the redundancy. If the syndrome is the all-zero vector, then the received word is a valid codeword (no errors). The non-zero syndromes have a one-to-one relationship with the error vectors provided the number of errors does not exceed the error-correcting capability of the code. (An error vector is a binary vector equal in length to an ECC codeword with the error positions having a value of 1 while the rest of the positions have the value 0). The *decoding* process is defined as the mapping of a syndrome to its associated error vector. Once an error vector is found, the corrected codeword can be calculated by XORing the error vector with the received word. For more background in error-correcting codes, the reader is referred to any book in the field, such as [2, 9].

ECC's differ in the number of errors which they can correct and also in the distance (measured as a Hamming distance in codespace) which can be recognized between the received word and a true codeword. Codes which can correct more errors and cover greater distances are considered more powerful. However, in practice the difficulty of developing an efficient decoder which can correct many errors prevents the use of most ECC's in the solution of real world problems. Although decoding can be done for any ECC via lookup table, this method quickly becomes intractable as the length of codewords and the number of errors possibly corrected increase. Development of an efficient decoder for a particular ECC is not straightforward. Moreover, it was shown that decoding of a random code is an NP-hard problem [1, 4].

The purpose of our work is to develop an ECC decoder using the trainable machine paradigm; i.e. we develop a decoder via training using examples of decoded received words. To prove our concept, we have selected a binary block code, the (23,12,7) Golay Code, which has "real world" complexity. The Golay Code corrects up to 3 errors and has minimum distance 7. A Golay codeword is 23 bits long (12 information bits, 11 bit redundancy); the syndrome is 11 bits long. There exist many efficient decoding methods for the Golay code [2, 3, 9], but the code complexity represents quite a challenge for our proposed approach.

## 1.2   A CONNECTIONIST ECC DECODER

We use a connectionist architecture as our ECC decoder; the input is a syndrome (we assume that the straightforward step of syndrome calculation is pre-processing) and the output is the portion of the error vector corresponding to the information bits in the received word (we ignore the redundancy). The primary reason for our choice of a connectionist architecture is its inherent simplicity and compactness; a connectionist architecture solution is readily implemented in either hardware or software solutions to complex real world problems. The particular architecture we use is the multi-layer feedforward network with one hidden layer. There are full connections only between adjacent layers. The number of nodes in the input layer is the number of bits in the syndrome, and the number of nodes in the output layer is the number of information bits in the ECC codeword. The number of nodes in the hidden layer is a free parameter, but typically this number is no more than 1 or 2 nodes greater than the number of nodes in the input layer. Our activation function is the logistic function and our training algorithm is backpropagation (see [10] for a description of both). This architectural approach has been demonstrated to be both cost-effective and a superior performer compared to classical statistical alternative methods in the solution of complex mapping problems when it is used as a trainable pattern classifier [6, 7].

There are two basic constraints which we have placed on our trainable connectionist decoder. First, the final connectionist architecture must be simple and contain as few nodes as possible. Second, the method we use to develop our decoder must be able to be generalized to any binary ECC. To meet the second constraint, we insured that the training dataset contained only examples of decoded words (i.e. no a priori knowledge of code patterning or existing decoding algorithms was included), and also that the training dataset was as small a subset of the possible error vectors as was required to obtain generalization by trained networks.

## 2   RESULTS

### 2.1   THE CASCADED NETWORKS DECODER

Using our basic approach, we have developed two separate solutions. One, the Cascaded Networks Decoder (see Figure 1) a system-level solution which parses the decoding problem into a set of more tractable problems each addressed by a separate network. These smaller networks each solve either simple classification problems (binary decisions) or are specialized decoders. Performance of the Cascaded Networks Decoder is 95% correct for the Golay code (tested on all $2^{11}$ possible error vectors), and the whole system is small and compact. However, this solution does not meet our constraint that the solution method be generic since the parsing of the original problem does require some a priori knowledge about the ECC, and the training of each network is done on a separate, self-contained schedule.

### 2.2   THE ECC-ENHANCED DECODER

The approach taken by the Cascaded Networks Decoder simplifies the solution strategy of the decoding problem, while the ECC-Enhanced Decoder simplifies the mapping problem to be solved by the decoder. In the ECC-Enhanced Decoder, both the input syndrome and the output error vector are encoded as codewords of an ECC. Such encoding should serve to separate the inputs in input space and the outputs in output space, creating a "region-to-region" mapping which is much easier than the "point-to-point" mapping required without encoding [8]. In addition, the decoding of the network output compensates for some level of uncertainty in the network's performance; an output vector within a small distance of the target vector will be corrected to the actual target by the ECC. This enhances training procedures [5, 8].

We have found that the ECC-Enhanced Decoder method meets all of our constraints for a connectionist architecture. However, we also have found that choosing the best ECC for encoding the input and for encoding the output represents two critical and quite separate problems which must be solved in order for the method to succeed.

#### 2.2.1   Choosing the Input ECC Encoding

The goal for the chosen ECC into which the input is encoded is to achieve maximum separation of input patterns in code space. The major constraint is the size of the codeword (number of bits which the length of the redundancy must be), because longer codewords increase the complexity of training and the size (in number of

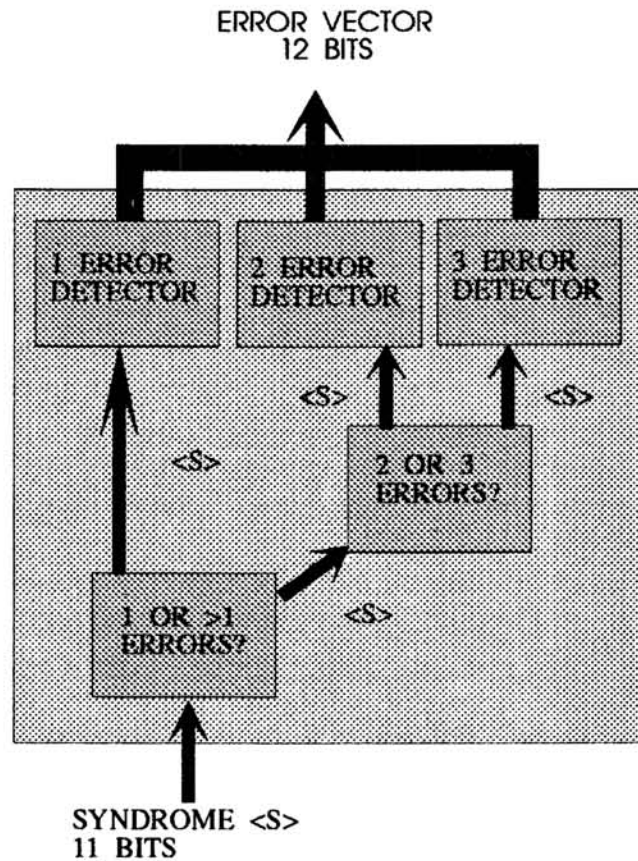

Figure 1: Cascaded Networks Decoder. A system-level solution incorporating 5 cascaded neural networks.

nodes) of the connectionist architecture. To determine the effect of different types of ECC's on the separation of input patterns in code space, we constructed a 325 pattern training dataset (mapping 11 bit syndrome to 12 bit error vector) and encoded only the inputs using 4 different ECC's. The candidate ECC's (with the size of redundancy required to encode the 11 bit syndrome) were

- Hamming (bit level, 4 bit redundancy)
- Extended Hamming (bit level, 5 bit redundancy)
- Reed Solomon (4 bit byte level, 2 byte redundancy)
- Fire (bit level, 11 bit redundancy)

We trained 5 networks (1 with no encoding of input, 1 each with a different ECC encoding) using this training dataset. Empirically, we had determined that this training dataset is slightly too small to achieve generalization for this task; we trained each network until its performance level on a 435 pattern test dataset (different patterns from the training dataset but encoded identically) degraded 20%. We then analyzed the effect of the input encoding on the patterning of error positions we observed for the output vectors.

The results of our analysis are illustrated in Figures 2 and 3. These bar graphs look only at output vectors found to have 2 or more errors, and show the proximity of error positions within an output vector. Each bar corresponds to the maximum distance of error positions within a vector (adjacent positions have a distance of 1). The bar height represents the total frequency of vectors with a given maximum distance; each bar is color-coded to break down the frequency by total number of errors per vector. This type of measurement shows the degree of burst (clustering of error positions) in the errors; knowing whether or not one has burst errors influences the likelihood of correction of those errors by an ECC (for instance, Fire codes are burst correcting codes).

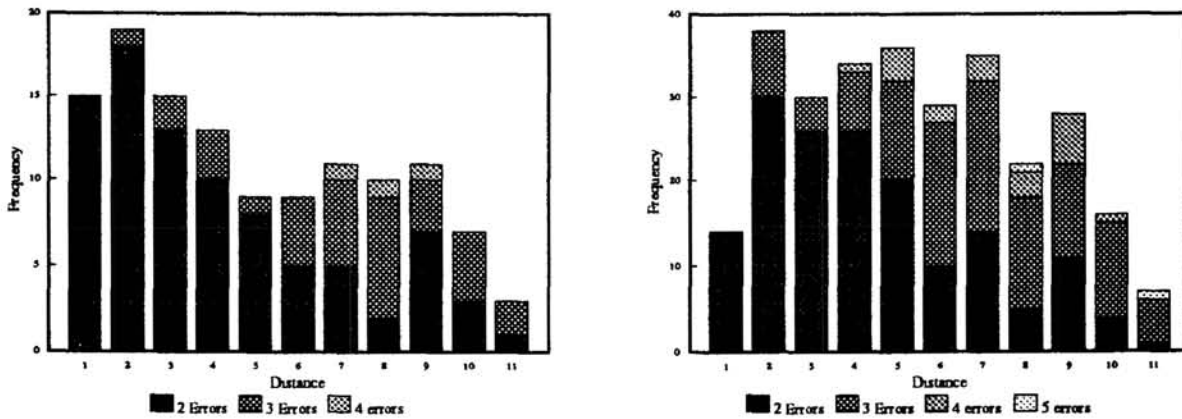

Figure 2: Bar Graphs of Output Errors Made by the Decoder. There was no encoding of the input in this instance. Training dataset results are on left, test dataset results are on right.

Our analysis shows that the Reed Solomon ECC is the only input encoding which separated the input patterns in a way which made use of an output pattern ECC encoding effective (resulted in more burst-type errors, decreased the total number of error positions in output vectors which had errors). The 11 bit redundancy required by the Fire code for input encoding increased complexity so that this solution was worse than the others in terms of performance. Thus, we have chosen the Reed Solomon ECC for input encoding in our ECC-Enhanced Decoder.

## 2.2.2   Choosing the Output ECC Encoding

The goal for the chosen ECC into which the output is encoded is correction of the maximum number of errors made by the decoder. Like the constraint imposed on the chosen ECC for input encoding, the ECC selected for encoding the output

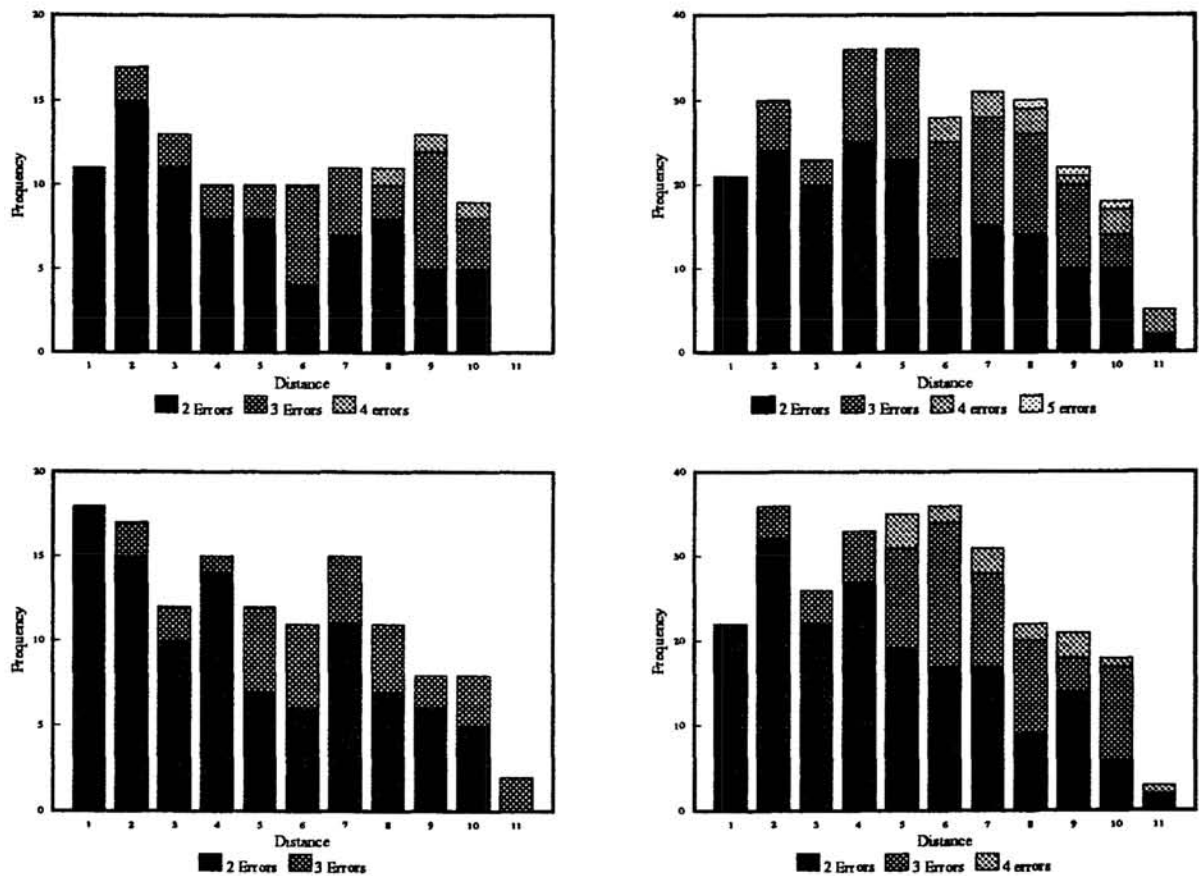

Figure 3: Bar Graphs of Effects of Different ECC Input Encodings on Output Errors Made by the Decoder. Training dataset results are on left, test dataset results are on right. Top row is Hamming code encoding, bottom row is Reed Solomon encoding.

should add as small a redundancy as possible. However, there is another even more important constraint on the choice of ECC for output encoding: decoding simplicity. The major advantage gained from encoding the output is the correction of slight uncertainty in the performance of the decoder, and this advantage is gained after the output is decoded. Thus, any ECC selected for output encoding should be one which can be decoded efficiently.

The error separation results we gained from our analysis of the effects of input encoding were used to guide our choices for an ECC into which the output would be encoded. We chose our ECC from the 4 candidates we considered for the input (these ECC's all can be decoded efficiently). The redundancy cost for encoding a 12 bit error vector was the same as in the 11 bit input case for the Reed Solomon and Fire codes, but was increased by 1 bit for the Hamming codes. Based on the result that a Reed Solomon encoding of the input both increased the amount of

burst errors and decreased the total number of errors per output vector, we chose the Hamming code and the Fire code for our output encoding ECC. Both encodings yielded excellent performance on the Golay code decoding problem; the Fire code output encoding resulted in better generalization by the network and thus better performance (87% correct) than the Hamming code output encoding (84% correct).

# References

[1] E. R. Berlekamp, R. J. McEliece and H. C. A. van Tilborg, "On the Inherent Intractability of Certain Coding Problems," *IEEE Trans. on Inf. Theory*, Vol. IT-8, pp. 384-386, May 1978.

[2] R. E. Blahut, *Theory and Practice of Error Control Codes*, Addison-Wesley, 1983.

[3] M. Blaum and J. Bruck, "Decoding the Golay Code with Venn Diagrams," *IEEE Trans. on Inf. Theory*, Vol. IT-36, pp. 906-910, July 1990.

[4] J. Bruck and M. Naor, "The Hardness of Decoding Linear Codes with Preprocessing," *IEEE Trans. on Inf. Theory*, Vol. IT-36, pp. 381-385, March 1990.

[5] T. G. Dietterich and G. Bakiri, "Error-Correcting Output Codes: A General Method for Improving Multiclass Inductive Learning Programs," Oregon State University Computer Science TR 91-30-2, 1991.

[6] S. L. Gish and W. E. Blanz, "Comparing a Connectionist Trainable Classifier with Classical Statistical Decision Analysis Methods," IBM Research Report RJ 6891 (65717), June 1989.

[7] S. L. Gish and W. E. Blanz, "Comparing the Performance of a Connectionist and Statistical Classifiers on an Image Segmentation Problem," in D. S. Touretzky (ed) *Neural Information Processing Systems 2*, pp. 614-621, Morgan Kaufmann Publishers, 1990.

[8] H. Li, T. Kronander and I. Ingemarsson, "A Pattern Classifier Integrating Multilayer Perceptron and Error-Correcting Code," in Proceedings of the IAPR Workshop on Machine Vision Applications, pp. 113-116, Tokyo, November 1990.

[9] F. J. MacWilliams and N. J. A. Sloane, *The Theory of Error-Correcting Codes*, Amsterdam, The Netherlands: North-Holland, 1977.

[10] D. E. Rumelhart, G. E. Hinton, and R. J. Williams, "Learning Internal Representations by Error Propagation," in D. E. Rumelhart, J. L. McClelland et. al. (eds) *Parallel Distributed Processing* Vol. 1, Chapter 8, MIT Press, 1986.
